# Learning the context of a category

**Daniel J. Navarro**
School of Psychology
University of Adelaide
Adelaide, SA 5005, Australia
daniel.navarro@adelaide.edu.au

## Abstract

This paper outlines a hierarchical Bayesian model for human category learning that learns both the organization of objects into categories, and the context in which this knowledge should be applied. The model is fit to multiple data sets, and provides a parsimonious method for describing how humans learn context specific conceptual representations.

## 1 Introduction

Human knowledge and expertise is often tied to particular contexts. The superior memory that chess masters have for chessboard configurations is limited to plausible games, and does not generalize to arbitrary groupings of pieces [1]. Expert firefighters make different predictions about the same fire depending on whether it is described as a back-burn or a to-be-controlled fire [2]. In part, this context specificity reflects the tendency for people to organize knowledge into independent "bundles" which may contain contradictory information, and which may be deemed appropriate to different contexts. This phenomenon is called *knowledge partitioning* [2–6], and is observed in artificial category learning experiments as well as real world situations. When people learn to classify stimuli in an environment where there are systematic changes in the "context" in which observations are made, they often construct category representations that are tightly linked to the context, and only generalize their knowledge when the context is deemed appropriate [3, 4, 6].

Context induced knowledge partitioning poses a challenge to models of human learning. As noted in [4] many models cannot accommodate the effect, or, as discussed later in this paper, are somewhat unsatisfying in the manner that they do so. This paper explores the possibility that Bayesian models of human category learning can provide the missing explanation. The structure of the paper is as follows: first, a context-sensitive Bayesian category learning model is described. This model is then shown to provide a parsimonious and psychologically appealing account of the knowledge partitioning effect. Following this, a hierarchical extension is introduced to the model, which allows it to acquire abstract knowledge about the context specificity of the categories, in a manner that is consistent with the data on human learning.

## 2 Learning categories in context

This section outlines a Bayesian model that is sensitive to the learning context. It extends Anderson's [7] rational model of categorization (RMC) by allowing the model to track the context in which observations are made, and draw inferences about the role that context plays.

### 2.1 The statistical model

The central assumption in the RMC is that the learner seeks to organize his or her observations into clusters. If $z_i$ denotes the cluster to which the $i$th observation is assigned, then the joint prior

distribution over $\mathbf{z}_n = (z_1, \ldots, z_n)$ can be specified via the Chinese restaurant process [8],

$$\mathbf{z}_n | \alpha \sim \mathrm{CRP}(\alpha). \tag{1}$$

Each cluster of observations is mapped onto a distribution over features. Feature values are denoted by the vector $\mathbf{x}_i = (x_{i1}, \ldots, x_{id})$, the values of the $i$th observation for each of the $d$ features. When feature values vary continuously, the RMC associates the $k$th cluster with a multivariate Gaussian that has mean vector $\boldsymbol{\mu}_k$ and covariance matrix $\boldsymbol{\Sigma}_k$. Setting standard conjugate priors, we obtain

$$
\begin{array}{rcll}
\mathbf{x}_i & | & \boldsymbol{\mu}_k, \boldsymbol{\Sigma}_k, z_i = k & \sim \quad \mathrm{Normal}(\boldsymbol{\mu}_k, \boldsymbol{\Sigma}_k) \\
\boldsymbol{\mu}_k & | & \boldsymbol{\Sigma}_k, \kappa_0, \boldsymbol{\mu}_0 & \sim \quad \mathrm{Normal}(\boldsymbol{\mu}_0, \boldsymbol{\Sigma}_k / \kappa_0) \\
\boldsymbol{\Sigma}_k & | & \boldsymbol{\Lambda}_0, \nu_0 & \sim \quad \mathrm{Inv\text{-}Wishart}(\nu_0, {\boldsymbol{\Lambda}_0}^{-1})
\end{array}
\tag{2}
$$

This is a minor generalization of the original model, as it allows any covariance matrix (i.e., symmetric positive definite $\boldsymbol{\Sigma}$) and does not require the restrictive assumption that the stimulus dimensions are independent (which would force $\boldsymbol{\Sigma}$ to be diagonal). While independence is reasonable when stimulus dimensions are separable [9], knowledge partitioning can occur regardless of whether dimensions are separable or integral (see [6] for details), so the more general formulation is useful.

In the RMC, labels are treated in the same way as discrete-valued features. Each cluster is associated with a distribution over category labels. If $\ell_i$ denotes the label given to the $i$th observation, then

$$
\begin{array}{rcll}
\ell_i & | & z_i = k, \theta_k & \sim \quad \mathrm{Bernoulli}(\theta_k) \\
\theta_k & | & \beta & \sim \quad \mathrm{Beta}(\beta, \beta)
\end{array}
\tag{3}
$$

The $\beta$ parameter describes the extent to which items in the same cluster are allowed to have different labels. If there are more than two labels, this generalizes to a Dirichlet-multinomial model.

Equations 1–3 define the standard RMC. The extension to handle context dependence is straightforward: contextual information is treated as an auxiliary feature, and so each cluster is linked to a distribution over contexts. In the experiments considered later, each observation is assigned to a context individually, which allows us to apply the exact same model for contextual features as regular ones. Thus a very simple context model is sufficient:

$$
\begin{array}{rcll}
c_i & | & z_i = k, \phi_k & \sim \quad \mathrm{Bernoulli}(\phi_k) \\
\phi_k & | & \gamma & \sim \quad \mathrm{Beta}(\gamma, \gamma)
\end{array}
\tag{4}
$$

The context specificity parameter $\gamma$ is analogous to $\beta$ and controls the extent to which clusters can include observations made in different contexts. In more general contexts, a richer model would be required to capture the manner in which context can vary.

Applying the model requires values to be chosen for $\alpha$, $\beta$, $\gamma$, $\boldsymbol{\mu}$, $\boldsymbol{\Lambda}_0$, $\nu_0$ and $\kappa_0$, most of which can be fixed in a sensible way. Firstly, since the categories do not overlap in the experiments discussed here it makes sense to set $\beta = 0$, which has the effect of forcing each cluster to be associated only with one category. Secondly, human learners rarely have strong prior knowledge about the features used in artificial category learning experiments, expressed by setting $\kappa_0 = 1$ and $\nu_0 = 3$ ($\nu_0$ is larger to ensure that the priors over features always has a well defined covariance structure). Thirdly, to approximate the fact that the experiments quickly reveal the full range of stimuli to participants, it makes sense to set $\boldsymbol{\mu}_0$ and $\boldsymbol{\Lambda}_0$ to the empirical mean and covariances across all training items. Having made these choices, we may restrict our attention to $\alpha$ (the bias to introduce new clusters) and $\gamma$ (the bias to treat clusters as context general).

## 2.2   Inference in the model

Inference is performed via a collapsed Gibbs sampler, integrating out $\phi$, $\theta$, $\boldsymbol{\mu}$ and $\boldsymbol{\Sigma}$ and defining a sampler only over the cluster assignments $\mathbf{z}$. To do so, note that

$$
\begin{array}{rcl}
P(z_i = k | \mathbf{x}, \boldsymbol{\ell}, \mathbf{c}, \mathbf{z}_{-i}) & \propto & P(\mathbf{x}_i, \ell_i, c_i | \mathbf{x}_{-i}, \boldsymbol{\ell}_{-i}, \mathbf{c}_{-i}, \mathbf{z}_{-i}, z_i = k) P(z_i = k | \mathbf{z}_{-i}) \qquad (5) \\
& = & P(\mathbf{x}_i | \mathbf{x}_{-i}, \mathbf{z}_{-i}, z_i = k) P(\ell_i | \boldsymbol{\ell}_{-i}, \mathbf{z}_{-i}, z_i = k) \\
& & P(c_i | \mathbf{c}_{-i}, \mathbf{z}_{-i}, z_i = k) P(z_i = k | \mathbf{z}_{-i}) \qquad\qquad\qquad (6)
\end{array}
$$

where the dependence on the parameters that describe the prior (i.e., $\alpha$, $\beta$, $\gamma$, $\boldsymbol{\Lambda}_0$, $\kappa_0$, $\nu_0$, $\boldsymbol{\mu}_0$) is suppressed for the sake of readability. In this expression $\mathbf{z}_{-i}$ denotes the set of all cluster assignments

except the $i$th, and the normalizing term is calculated by summing Equation 6 over all possible cluster assignments $k$, including the possibility that the $i$th item is assigned to an entirely new cluster.

The conditional prior probability $P(z_i = k | \mathbf{z}_{-i})$ is

$$P(z_i = k | \mathbf{z}_{-i}) = \begin{cases} \frac{n_k}{n-1+\alpha} & \text{if } k \text{ is old} \\ \frac{\alpha}{n-1+\alpha} & \text{if } k \text{ is new} \end{cases} \tag{7}$$

where $n_k$ counts the number of items (not including the $i$th) that have been assigned to the $k$th cluster. Since the context is modelled using a beta-Bernoulli model:

$$P(c_i | \mathbf{c}_{-i}, \mathbf{z}_{-i}, z_i = k) = \int_0^1 P(c_i | \phi_k, z_i = k) P(\phi_k | \mathbf{c}_{-i}, \mathbf{z}_{-i}) \, d\phi_k = \frac{n_k^{(c_i)} + \gamma}{n_k + 2\gamma} \tag{8}$$

where $n_k^{(c_i)}$ counts the number of observations that have been assigned to cluster $k$ and appeared in the same context as the $i$th item. A similar result applies to the labelling scheme:

$$P(\ell_i | \boldsymbol{\ell}_{-i}, \mathbf{z}_{-i}, z_i = k) = \int_0^1 P(\ell_i | \theta_k, z_i = k) P(\theta_k | \boldsymbol{\ell}_{-i}, \mathbf{z}_{-i}) \, d\theta_k = \frac{n_k^{(\ell_i)} + \beta}{n_k + 2\beta} \tag{9}$$

where $n_k^{(\ell_i)}$ counts the number of observations that have been assigned to cluster $k$ and given the same label as observation $i$. Finally, integrating out the mean vector $\boldsymbol{\mu}_k$ and covariance matrix $\boldsymbol{\Sigma}_k$ for the feature values yields a $d$-dimensional multivariate $t$ distribution (e.g., [10], ch. 3):

$$P(\mathbf{x}_i | \mathbf{x}_{-i}, \mathbf{z}_{-i}, z_i = k) = \int P(\mathbf{x}_i | \boldsymbol{\mu}_k, \boldsymbol{\Sigma}_k, z_i = k) P(\boldsymbol{\mu}_k, \boldsymbol{\Sigma}_k | \mathbf{x}_{-i}, \mathbf{z}_{-i}) \, d(\boldsymbol{\mu}_k, \boldsymbol{\Sigma}_k) \tag{10}$$

$$= \frac{\Gamma(\frac{\nu'_k + d}{2})}{\Gamma(\frac{\nu'_k}{2})(\pi \nu'_k)^{\frac{d}{2}} |\boldsymbol{\Lambda}'_k|^{\frac{1}{2}}} \left( 1 + \frac{(\mathbf{x}_i - \boldsymbol{\mu}'_k) \boldsymbol{\Lambda}'_k{}^{-1} (\mathbf{x}_i - \boldsymbol{\mu}'_k)^{\mathsf{T}}}{\nu'_k} \right)^{-\frac{\nu'_k + d}{2}} \tag{11}$$

In this expression the posterior degrees of freedom for cluster $k$ is $\nu'_k = \nu_0 + n_k - d + 1$ and the posterior mean is $\boldsymbol{\mu}'_k = (\kappa_0 \boldsymbol{\mu}_0 + n_k \bar{\mathbf{x}}_k)/(\kappa_0 + n_k)$, where $\bar{\mathbf{x}}_k$ denotes the empirical mean feature values for items in the cluster. Finally, the posterior scale matrix is

$$\boldsymbol{\Lambda}'_k = \left( \boldsymbol{\Lambda}_0 + \mathbf{S}_k + \frac{\kappa_0 n_k}{\kappa_0 + n_k} (\bar{\mathbf{x}}_k - \boldsymbol{\mu}_0)^{\mathsf{T}} (\bar{\mathbf{x}}_k - \boldsymbol{\mu}_0) \right) \frac{\kappa_0 + n_k + 1}{(\kappa_0 + n_k)(\nu_0 + n_k - 2d + 2)} \tag{12}$$

where $\mathbf{S}_k = \sum (\mathbf{x}_i - \bar{\mathbf{x}}_k)^{\mathsf{T}} (\mathbf{x}_i - \bar{\mathbf{x}}_k)$ is the sum of squares matrix around the empirical cluster mean $\bar{\mathbf{x}}_k$, and the sum in question is taken over all observations assigned to cluster $k$.

Taken together, Equations 6, 8, 9 and 11 suggest a simple a Gibbs sampler over the cluster assignments $\mathbf{z}$. Cluster assignments $z_i$ are initialized randomly, and are then sequentially redrawn from the conditional posterior distribution in Equation 6. For the applications in this paper, the sampler typically converges within only a few iterations, but a much longer burn in (usually 1000 iterations, never less than 100) was used in order to be safe. Successive samples are drawn at a lag of 10 iterations, and multiple runs (between 5 and 10) are used in all cases.

## 3 Application to knowledge partitioning experiments

To illustrate the behavior of the model, consider the most typical example of a knowledge partitioning experiment [3, 4, 6]. Stimuli vary along two continuous dimensions (e.g., height of a rectangle, location of a radial line), and are organized into categories using the scheme shown in Figure 1a. There are two categories organized into an "inside-outside" structure, with one category (black circles/squares) occupying a region along either side of the other one (white circles/squares). The critical characteristic of the experiment is that each stimulus is presented in a particular "context", usually operationalized as an auxiliary feature not tied to the stimulus itself, such as the background color. In Figure 1a, squares correspond to items presented in one context, and circles to items presented in the other context. Participants are trained on these items in a standard supervised categorization experiment: stimuli are presented one at a time (with the context variable), and participants are asked to predict the category label. After making a prediction, the true label is revealed to them.

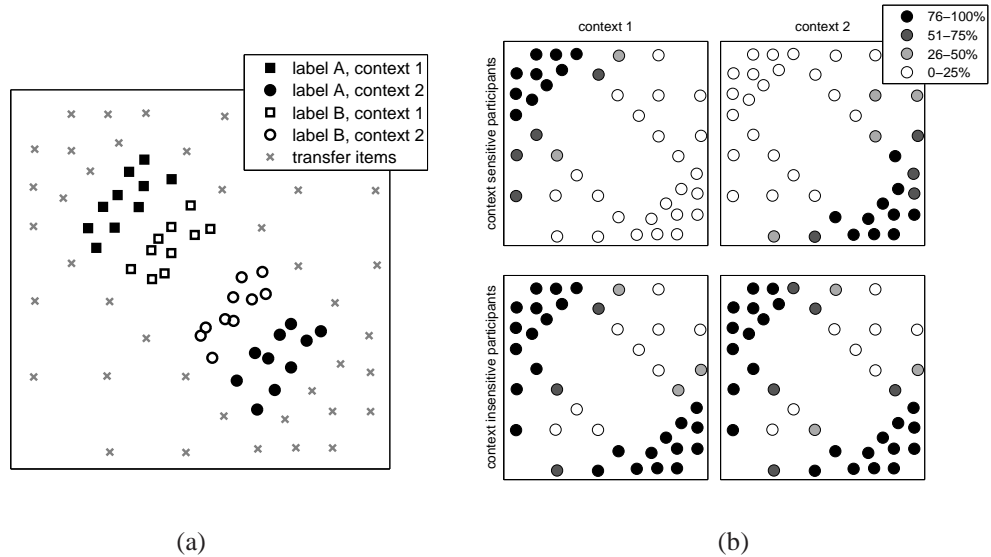

(a)                                                                                        (b)

Figure 1: Stimuli used in the typical knowledge partitioning design (left) and the different generalization patterns that are displayed by human learners (right). Percentages refer to the probability of selecting category label A.

This procedure is repeated until participants can correctly label all items. At this point, participants are shown transfer items (the crosses in Figure 1a), and asked what category label these items should be given. No feedback is given during this phase. Critically, each transfer item is presented in both contexts, to determine whether people generalize in a context specific way.

The basic effect, replicated across several different experiments, is that there are strong individual differences in how people solve the problem. This leads to the two characteristic patterns of generalization shown in Figure 1b (these data are from Experiments 1 and 2A in [6]). Some participants are context insensitive (lower two panels) and their predictions about the transfer items do not change as a function of context. However, other participants are context sensitive (upper panels) and adopt a very different strategy depending on which context the transfer item is presented in. This is taken to imply [3, 4, 6] that the context sensitive participants have learned a conceptual representation in which knowledge is "partitioned" into different bundles, each associated with a different context.

## 3.1 Learning the knowledge partition

The initial investigation focused on what category representations the model learns, as a function of $\alpha$ and $\gamma$. After varying both parameters over a broad range, it was clear that there are two quite different solutions that the model can produce, illustrated in Figure 2. In the four cluster solution (panel b, small $\gamma$), the clusters never aggregate across items observed in different contexts. In contrast, the three cluster solution (panel a, larger $\gamma$) is more context general, and collapses category B into a single cluster. However, there is an interaction with $\alpha$, since large $\alpha$ values drive the model to introduce more clusters. As a result, for $\alpha > 1$ the model tends not to produce the three cluster solution. Given that the main interest is in $\gamma$, we can fix $\alpha$ such that the prior expected number of clusters is 3.5, so as to be neutral with respect to the two solutions. Since the expected number of clusters is given by $\alpha \sum_{k=0}^{n-1}(\alpha + k)$ [11] and there are $n = 40$ observations, this value is $\alpha = 0.72$.

The next aim was to quantify the extent to which $\gamma$ influences the relative prevalence of the four cluster solution versus the three cluster solution. For any given partition produced by the model, the adjusted Rand index [12] can be used to assess its similarity to the two idealized solutions (Figure 2a and 2b). Since the adjusted Rand index measures the extent to which any given pair of items are classified in the same way by the two solutions, it is a natural measure of how close a model-generated solution is to one of the two idealized solutions. Then, adopting an approach loosely inspired by PAC-learning [13], two partitions were deemed to be approximately the same if the adjusted Rand

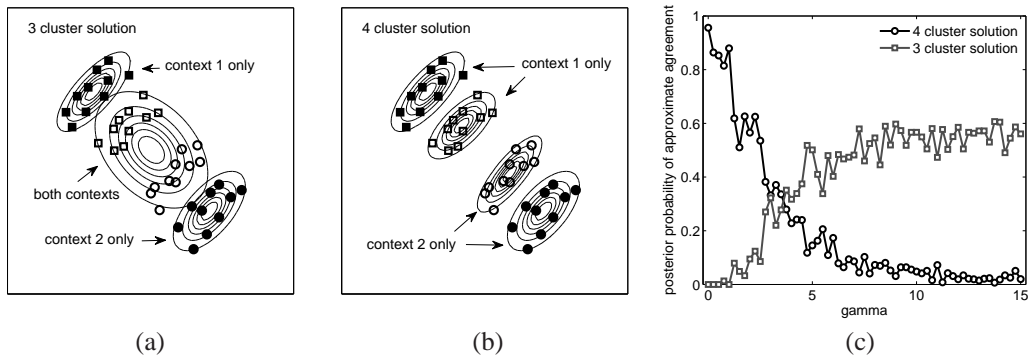

Figure 2: The two different clustering schemes produced by the context sensitive RMC, and the values of $\gamma$ that produce them (for $\alpha$ fixed at 0.72). See main text for details.

index between the two exceeded 0.9. The estimated posterior probability that the model solutions approximate either of the the two idealized partitions is plotted in Figure 2c as a function of $\gamma$. At smaller values of $\gamma$ (below about 3.7) the four cluster solution is extremely dominant whereas at larger values the three cluster solution is preferred. Since there are approximately $1.6 \times 10^{35}$ possible partitions of 40 objects, the extent of this dominance is clearly very strong.

The fact that the model concentrates on two different but entirely sensible solutions as a function of $\gamma$ is very appealing from a psychological perspective. One of the most desirable characteristics is the fact that the partitioning of the learners knowledge is made explicit. That is, the model learns a much more differentiated and context bound representation when $\gamma$ is small, and a more context general and less differentiated representation when $\gamma$ is large. By way of comparison, the only other model that has been shown to produce the effect is ATRIUM [14], which in its standard form consists of a linked "rule learning" module and an "exemplar learning" module. In order to fit the data, the model was modified [4] so that it starts with two rule modules and an exemplar model. During training, the model learns to weight each of the rule modules differently depending on context, thereby producing context specific generalizations. This provides a partial explanation of the effect, but it is rather unsatisfing in some ways. In ATRIUM, the knowledge partition is represented via the learned division of responsibilities between two hard coded rule modules [4]. In a very real sense, the partition is actually hard coded into the architecture of the model. As such, ATRIUM learns the context dependence, but not the knowledge partition itself.

## 3.2 Generalizing in context-specific and context-general ways

The discussion to this point shows how the value of $\gamma$ shapes the conceptual knowledge that the model acquires, but has not looked at what generalizations the model makes. However, it is straightforward to show that varying $\gamma$ does allow the context sensitive RMC to capture the two generalization patterns in Figure 1. With this in mind, Figure 3 plots the generalizations made by the model for two different levels of context specificity ($\gamma = 0$ and $\gamma = 10$) and for the two different clustering solutions. Obviously, in view of the results in Figure 2c the most interesting cases are panels (a) and (d), since those correspond to the solutions most likely to be learned by the model, but it is useful to consider all four cases. As is clear from inspection – and verified by the squared correlations listed in the Figure caption – when $\gamma$ is small the model generalizes in a context specific manner, but when $\gamma$ is large the generalizations are the same in all contexts. This happens for both clustering solutions, which implies that $\gamma$ plays two distinct but related roles, insofar as it influences the context specificity of *both* the learned knowledge partition and the generalizations to new observations.

## 4 Acquiring abstract knowledge about context specificity

One thing missing from both ATRIUM and the RMC is an explanation for how the leaner decides whether context specific or context general representations are appropriate. In both cases, the model has free parameters that govern the switch between the two cases, and these parameters must be

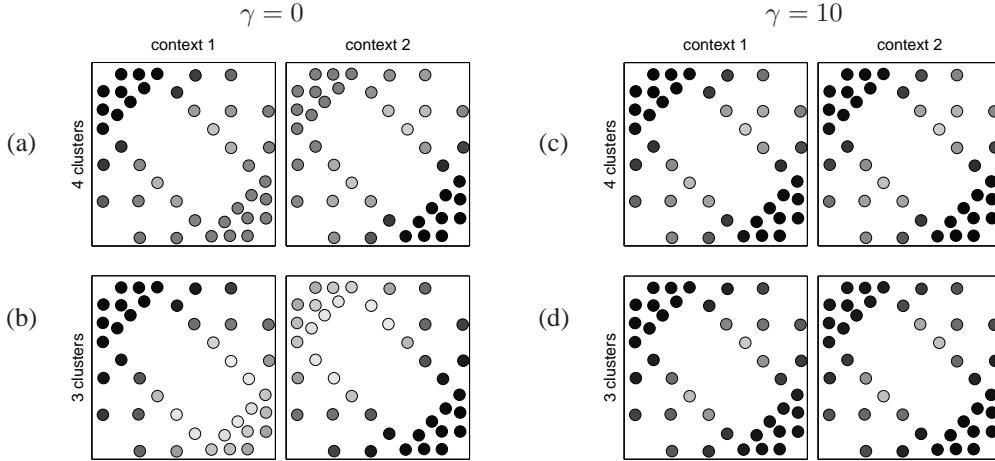

Figure 3: Generalizations made by the model. In panel (a) the model accounts for 82.1% of the variance in the context sensitive data, but only 35.2% of the variance in the context insensitive data. For panel (b) these numbers are 77.9% and 3.6% respectively. When $\gamma$ is large the pattern reverses: in panel (c) only 23.6% of the variance in the context sensitive data is explained, whereas 67.1% of the context insensitive data can be accounted for. In panel (d), the numbers are 17.5% and 73.9%.

estimated from data. In the RMC, $\gamma$ is a free parameter that does all the work; for ATRIUM, four separate parameters are varied [4]. This poses the question: how do people acquire abstract knowledge about which way to generalize? In RMC terms, how do we infer the value of $\gamma$?

To answer this, note that if the context varies in a systematic fashion, an intelligent learner might come to suspect that the context matters, and would be more likely to decide to generalize in a context specific way. On the other hand, if there are no systematic patterns to the way that observations are distributed across contexts, then the learner should deem the context to be irrelevant and hence decide to generalize broadly across contexts. Indeed, this is exactly what happens with human learners. For instance, consider the data from Experiment 1 in [4]. One condition of this experiment was a standard knowledge partitioning experiment, identical in every meaningful respect to the data described earlier in this paper. As is typical for such experiments, knowledge partitioning was observed for at least some of the participants. In the other condition, however, the context variable was randomized: each of the training items was assigned to a randomly chosen context. In this condition, no knowledge partitioning was observed.

What this implies is that human learners use the systematicity of the context as a cue to determine how broadly to generalize. As such, the model should *learn* that $\gamma$ is small when the context varies systematically; and similarly should learn that $\gamma$ is large if the context is random. To that end, this section develops a hierarchical extension to the model that is able to do exactly this, and shows that it is able to capture both conditions of the data in [4] without varying any parameter values.

## 4.1 A hierarchical context-sensitive RMC

Extending the statistical model is straightforward: we place priors over $\gamma$, and allow the model to infer a joint posterior distribution over the cluster assignments $\mathbf{z}$ and the context specificity $\gamma$. This is closely related to other hierarchical Bayesian models of category learning [15–19]. A simple choice of prior for this situation is the exponential distribution,

$$\gamma | \lambda \sim \text{Exponential}(\lambda) \qquad (13)$$

Following the approach taken with $\alpha$, $\lambda$ was fixed so as to ensure that the model has no a priori bias to prefer either of the two solutions. When $\gamma = 3.7$ the two solutions are equally likely (Figure 2); a value of $\lambda = .19$ ensures that this value of $\gamma$ is the prior median.

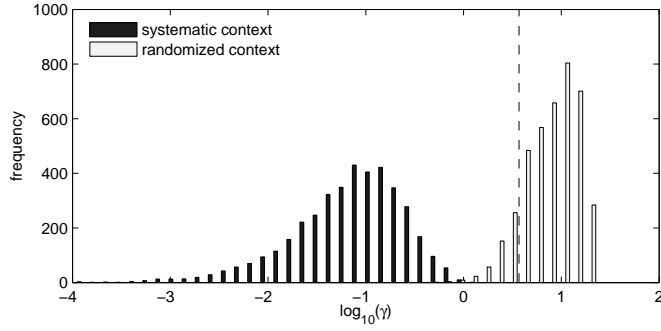

Figure 4: Learned distributions over $\gamma$ in the systematic (dark rectangles) and randomized (light rectangles) conditions, plotted on a logarithmic scale. The dashed line shows the location of the prior median (i.e., $\gamma = 3.7$).

Inference in the hierarchical model proceeds as before, with a Metropolis step added to resample $\gamma$. The acceptance probabilities for the Metropolis sampler may be calculated by observing that

$$
\begin{aligned}
P(\gamma|\mathbf{x}, \boldsymbol{\ell}, \mathbf{c}, \mathbf{z}) &\propto P(\mathbf{x}, \boldsymbol{\ell}, \mathbf{c}|\mathbf{z}, \gamma)P(\gamma) && (14) \\
&\propto P(\mathbf{c}|\mathbf{z}, \gamma)P(\gamma) && (15) \\
&= \int P(\mathbf{c}|\mathbf{z}, \boldsymbol{\phi})P(\boldsymbol{\phi}|\gamma) \, d\boldsymbol{\phi} \; P(\gamma) && (16) \\
&= P(\gamma) \prod_{k=1}^{K} \int_0^1 P(\mathbf{c}^{(k)}|\phi_k)P(\phi_k|\gamma) \, d\phi_k && (17) \\
&= \lambda \exp(-\lambda\gamma) \prod_{k=1}^{K} \frac{n_k!}{n_k^{(c=1)}! n_k^{(c=2)}!} \frac{B(n_k^{(c=1)} + \gamma, n_k^{(c=2)} + \gamma)}{B(\gamma, \gamma)} && (18) \\
&\propto \exp(-\lambda\gamma) \prod_{k=1}^{K} \frac{B(n_k^{(c=1)} + \gamma, n_k^{(c=2)} + \gamma)}{B(\gamma, \gamma)} && (19)
\end{aligned}
$$

where $B(a,b) = \Gamma(a)\Gamma(b)/\Gamma(a+b)$ denotes the beta function, and $n_k^{(c=j)}$ counts the number of items in cluster $k$ that appeared in context $j$.

## 4.2 Application of the extended model

To explore the performance of the hierarchical extension of the context sensitive RMC, the model was trained on both the original, systematic version of the knowledge partitioning experiments, and on a version with the context variables randomly permuted. The posterior distributions over $\gamma$ that this produces are shown in Figure 4. As expected, in the systematic condition the model notices the fact that the context varies systematically as a function of the feature values $\mathbf{x}$, and learns to form context specific clusters. Indeed, 97% of the posterior distribution over $\mathbf{z}$ is absorbed by the four cluster solution (or other solutions that are sufficiently similar in the sense discussed earlier). In the process, the model infers that $\gamma$ is small and generalizes in a context specific way (as per Figure 3). Nevertheless, without changing any parameter values, the same model in the randomized condition infers that there is no pattern to the context variable, which ends up being randomly scattered across the clusters. For this condition 57% of the posterior mass is approximately equivalent to the three cluster solution. As a result, the model infers that $\gamma$ is large, and generalizes in the context general fashion. In short, the model captures human performance quite effectively.

When considering the implications of Figure 4, it is clear that the model captures the critical feature of the experiment: the ability to *learn* when to make context specific generalizations and when not to. The distributions over $\gamma$ are very different as a function of condition, indicating that the model learns appropriately. What is less clear is the extent to which the model would be expected to produce the correct pattern of individual differences. Inspection of Figure 4 reveals that in the

randomized context condition the posterior distribution over $\gamma$ does not move all that far above the prior median of 3.7 (dashed line) which by construction is intended to be a fairly neutral value, whereas in the systematic condition nearly the entire distribution lies below this value. In other words, the systematic condition produces more learning about $\gamma$. If one were to suppose that people had no inherent prior biases to prefer to generalize one way or the other, it should follow that the less informative condition (i.e., random context) should reveal more individual differences. Empirically, the reverse is true: in the less informative condition, all participants generalize in a context general fashion; whereas in the more informative condition (i.e., systematic context) some but not all participants learn to generalize more narrowly. This does not pose any inherent difficulty for the model, but it does suggest that the "unbiased" prior chosen for this demonstration is not quite right: people do appear to have strong prior biases to prefer context general representations. Fortunately, a cursory investigation revealed that altering the prior over $\gamma$ moves the posteriors in a sensible fashion while still keeping the two distributions distinct.

## 5  Discussion

The hierarchical Bayesian model outlined in this paper explains how human conceptual learning can be context general in some situations, and context sensitive in others. It captures the critical "knowledge partitioning" effect [2–4, 6] and does so without altering the core components of the RMC [7] and its extensions [15, 16, 18, 20]. This success leads to an interesting question: why does ALCOVE [21] *not* account for knowledge partitioning (see [4])? Arguably, ALCOVE has been the dominant theory for learned selective attention for almost 20 years, and its attentional learning mechanisms bear a striking similarity to the hierarchical Bayesian learning idea used in this paper and elsewhere [15–19], as well as to statistical methods for automatic relevance determination in Bayesian neural networks [22]. On the basis of these similarities, one might expect similar behavior from ALCOVE and the context sensitive RMC. Yet this is not the case. The answer to this lies in the details of *why* one learns dimensional biases. In ALCOVE, as in many connectionist models, the dimensional biases are chosen to optimize the ability to predict the category label. Since the context variable is not correlated with the label in these experiments (by construction), ALCOVE learns to ignore the context variable in all cases. The approach taken by the RMC is qualitatively different: it looks for clusters of items where the label, the context and the feature values are all similar to one another. Knowledge partitioning experiments more or less require that such clusters exist, so the RMC can learn that the context variable is not distributed randomly. In short, ALCOVE treats context as important only if it can predict the label; the RMC treats the context as important if it helps the learner infer the structure of the world.

Looking beyond artificial learning tasks, learning the situations in which knowledge should be applied is an important task for an intelligent agent operating in a complex world. Moreover, hierarchical Bayesian models provide a natural formalism for describing how human learners are able to do so. Viewed in this light, the fact that it is possible for people to hold contradictory knowledge in different "parcels" should be viewed as a special case of the general problem of learning the set of relevant contexts. Consider, for instance, the example in which fire fighters make different judgments about the same fire depending on whether it is called a back-burn or a to-be-controlled fire [2]. If fire fighters observe a very different distribution of fires in the context of back-burns than in the context of to-be-controlled fires, then it should be no surprise that they acquire two distinct theories of "fires", each bound to a different context. Although this particular example is a case in which the learned context specificity is incorrect, it takes only a minor shift to make the behavior correct. While the behavior of fires does not depend on the reason why they were lit, it does depend on what combustibles they are fed. If the distinction were between fires observed in a forest context and fires observed in a tyre yard, context specific category representations suddenly seem very sensible. Similarly, social categories such as "polite behavior" are necessarily highly context dependent, so it makes sense that the learner would construct different rules for different contexts. If the world presents the learner with observations that vary systematically across contexts, partitioning knowledge by context would seem to be a rational learning strategy.

**Acknowledgements**

This research was supported by an Australian Research Fellowship (ARC grant DP-0773794).

# References

[1] W. G. Chase and H. A. Simon. Perception in chess. *Cognitive Psychology*, 4:55–81, 1973.

[2] S. Lewandowsky and K. Kirsner. Knowledge partitioning: Context-dependent use of expertise. *Memory and Cognition*, 28:295–305, 2000.

[3] L.-X. Yang and S. Lewandowsky. Context-gated knowledge partitioning in categorization. *Journal of Experimental Psychology: Learning, Memory, and Cognition*, 29:663–679, 2003.

[4] L.-X. Yang and S. Lewandowsky. Knowledge partitioning in categorization: Constraints on exemplar models. *Journal of Experimental Psychology: Learning, Memory, and Cognition*, 30:1045–1064, 2004.

[5] M. L. Kalish, S. Lewandowsky, and J. K. Kruschke. Population of linear experts: Knowledge partitioning in function learning. *Psychological Review*, 111:1072–1099, 2004.

[6] S. Lewandowsky, L. Roberts, and L.-X. Yang. Knowledge partitioning in category learning: Boundary conditions. *Memory and Cognition*, 38:1676–1688, 2006.

[7] J. R. Anderson. The adaptive nature of human categorization. *Psychological Review*, 98: 409–429, 1991.

[8] D. Aldous. Exchangeability and related topics. In *École d'été de probabilités de Saint-Flour, XIII-1983*, pages 1–198. Springer, Berlin, 1985.

[9] R. N. Shepard. Integrality versus separability of stimulus dimensions: From an early convergence of evidence to a proposed theoretical basis. In J. R. Pomerantz and G. L. Lockhead, editors, *The Perception of Structure: Essays in Honor of Wendell R. Garner*, pages 53–71. American Psychological Association, Washington, DC, 1991.

[10] A. Gelman, J. B. Carlin, H. S. Stern, and D. B. Rubin. *Bayesian Data Analysis*. Chapman and Hall, Boca Raton, 2nd edition, 2004.

[11] C. E. Antoniak. Mixtures of Dirichlet processes with applications to Bayesian nonparametric problems. *Annals of Statistics*, 2:1152–1174, 1974.

[12] L. Hubert and P. Arabie. Comparing partitions. *Journal of Classification*, 2:193–218, 1985.

[13] L. Valiant. A theory of the learnable. *Communications of the ACM*, 27:1134–1142, 1984.

[14] M. A. Erickson and J. K. Kruschke. Rules and exemplars in category learning. *Journal of Experimental Psychology: General*, 127:107–140, 1998.

[15] C. Kemp, A. Perfors, and J. B. Tenenbaum. Learning overhypotheses with hierarchical Bayesian models. *Developmental Science*, 10:307–332, 2007.

[16] A. Perfors and J. B. Tenenbaum. Learning to learn categories. In N. Taatgen, H. van Rijn, L. Schomaker, and J. Nerbonne, editors, *Proceedings of the 31st Annual Conference of the Cognitive Science Society*, pages 136–141, Austin, TX, 2009. Cognitive Science Society.

[17] D. J. Navarro. From natural kinds to complex categories. In R. Sun and N. Miyake, editors, *Proceedings of the 28th Annual Conference of the Cognitive Science Society*, pages 621–626, Mahwah, NJ, 2006. Lawrence Erlbaum.

[18] T. L. Griffiths, K. R. Canini, A. N. Sanborn, and D. J. Navarro. Unifying rational models of categorization via the hierarchical Dirichlet process. In D. S. McNamara and J. G. Trafton, editors, *Proceedings of the 29th Annual Conference of the Cognitive Science Society*, pages 323–328, Austin, TX, 2007. Cognitive Science Society.

[19] K. Heller, A. N. Sanborn, and N. Chater. Hierarchical learning of dimensional biases in human categorization. In J. Lafferty and C. Williams, editors, *Advances in Neural Information Processing Systems 22*, Cambridge, MA, 2009. MIT Press.

[20] A. N. Sanborn, T. L. Griffiths, and D. J. Navarro. Rational approximations to rational models: Alternative algorithms for category learning. *Psychological Review*, in press.

[21] J. K. Kruschke. ALCOVE: An exemplar-based connectionist model of category learning. *Psychological Review*, 99:22–44, 1992.

[22] R. Neal. *Bayesian learning for neural networks*. Springer-Verlag, New York, 1996.

